# Data Skeletonization via Reeb Graphs

**Xiaoyin Ge**          **Issam Safa**          **Mikhail Belkin**          **Yusu Wang**

Computer Science and Engineering Department
The Ohio State University
`gex,safa,mbelkin,yusu@cse.ohio-state.edu`

## Abstract

Recovering hidden structure from complex and noisy non-linear data is one of the most fundamental problems in machine learning and statistical inference. While such data is often high-dimensional, it is of interest to approximate it with a low-dimensional or even one-dimensional space, since many important aspects of data are often intrinsically low-dimensional. Furthermore, there are many scenarios where the underlying structure is graph-like, e.g, river/road networks or various trajectories. In this paper, we develop a framework to extract, as well as to simplify, a one-dimensional "skeleton" from unorganized data using the Reeb graph. Our algorithm is very simple, does not require complex optimizations and can be easily applied to unorganized high-dimensional data such as point clouds or proximity graphs. It can also represent arbitrary graph structures in the data. We also give theoretical results to justify our method. We provide a number of experiments to demonstrate the effectiveness and generality of our algorithm, including comparisons to existing methods, such as principal curves. We believe that the simplicity and practicality of our algorithm will help to promote skeleton graphs as a data analysis tool for a broad range of applications.

## 1   Introduction

Learning or inferring a hidden structure from discrete samples is a fundamental problem in data analysis, ubiquitous in a broad range of application fields. With the rapid generation of diverse data all across science and engineering, extracting geometric structure is often a crucial first step towards interpreting the data at hand, as well as the underlying process of phenomenon. Recently, there has been a large amount of research in this direction, especially in the machine learning community.

In this paper, we consider a simple but important scenario, where the hidden space has a *graph-like geometric structure*, such as the branching filamentary structures formed by blood vessels. Our goal is to extract such structures from points sampled on and around them. Graph-like geometric structures arise naturally in many fields, both in modeling natural phenomena, and in understanding abstract procedures and simulations. However, there has been only limited work on obtaining a general-purpose algorithm to automatically extract skeleton graph structures [2]. In this paper, we present such an algorithm by bringing in a topological concept called the Reeb graph to extract skeleton graphs. Our algorithm is simple, efficient and easy to use. We demonstrate the generality and effectiveness of our algorithm via several applications in both low and high dimensions.

**Motivation.** Geometric graphs are the underlying structures for modeling many natural phenomena from river / road networks, root systems of trees, to blood vessels, and particle trajectories. For example, if we are interested in obtaining the road network of a city, we may send out cars to explore various streets of the city, with each car recording its position using a GPS device. The resulting data is a set of potentially noisy points sampled from the roads in a city. Given these data, the goal is to *automatically reconstruct* the road network, which is a *graph* embedded in a two- dimensional space. Indeed, abundant data of this type are available at the open-streets project website [1].

Geometric graphs also arise from many modeling processes, such as molecular simulations. They can sometimes provide a natural platform to study a collection of time-series data, where each time-series corresponds to a trajectory in the feature space. These trajectories converge and diverge, which can be represented by a graph. This graph in turn can then be used as a starting point for further processing (such as matching) or inference tasks.

Generally, there are a number of scenarios where we wish to extract a one-dimensional skeleton from an input space. The goal in this paper is to develop, as well as to demonstrate the use of, a practical and general algorithm to extract a graph structure from input data of any dimensions.

**New work.** Given a set of points $P$ sampling a hidden domain $X$, we present a simple and practical algorithm to extract a skeleton graph $G$ for $X$. The input points $P$ do not have to be embedded – we only need their distance matrix or simply a proximity graph as input to our algorithm.

Our algorithm is based on using the so-called Reeb graph to model skeleton graphs. Given a continuous function $f : X \to \mathbb{R}$, the Reeb graph tracks the connected components in the level-set $f^{-1}(a)$ of $f$ as we vary the value $a$. It provides a meaningful abstraction of the scalar field $f$, and has been widely used in graphics, visualization, and computer vision (see [6] for a survey). However, it has not yet been aimed as a tool to analyze high dimensional data from unorganized input data. By bringing the concept of the Reeb graph to machine learning applications, we can leverage the recent algorithms developed to compute and process Reeb graphs [15, 9]. Moreover, combining the Reeb graph with the so-called Rips complex allows us to obtain theoretical guarantees for our algorithm.

Our algorithm is simple and efficient. There is only one parameter involved, which intuitively specifies the scale at which we look at the data. Our algorithm always outputs a graph $G$ given data. Furthermore, it also computes a map $\Phi : P \to G$, which maps each sample point to $G$. Hence we can decompose the input data into sets, each corresponding to a single branch in the skeleton graph. Finally, there is a canonical way to measure importance of features in the Reeb graph, which allows us to easily simplify the resulting graph. We summarize our contributions as follows:

(1) We bring in Reeb graphs to the learning community for analyzing high dimensional unorganized data sets. We developed an accompanying software to not only extract, but also process skeleton graphs from data. Our algorithm is simple and robust, always extracting a graph from the input. Our algorithm complements principal curve algorithms and can be used in combination with them.

(2) We provide certain theoretical guarantees for our algorithm. We also demonstrate both the effectiveness of our software and the usefulness of skeleton graphs via a sets of experiments on diverse datasets. Experimental results show that despite being simple and general, our algorithm compares favorably to existing graph-extracting algorithms in various settings.

**Related work.** At a broad level, the graph-extraction problem is related to manifold learning and non-linear dimensionality reduction which has a rich literature, see e.g [4, 24, 25, 27]. Manifold learning methods typically assume that the hidden domain has a manifold structure. An even more general scenario is that the hidden domain is a *stratified space*, which intuitively, can be thought of as a collection of manifolds (strata) glued together. Recently, there have been several approaches to learn stratified spaces [5, 14]. However, this general problem is hard and requires algorithms both mathematically sophisticated and computationally intensive. In this case, we aim to learn a graph structure, which is simply a one-dimensional stratified space, allowing for simple approaches.

The most relevant previous work related to our graph-extraction problem is a series of results on an elegant concept of *principal curves*, originally proposed by Hastie and Stuetzle [16, 17]. Intuitively, principal curves are "self-consistent" curves that pass through the middle of the data. Since its original introduction, there has been much work on analyzing and extending the concept and algorithms as well as on numerous applications. See, e.g, [7, 11, 10, 19, 22, 26, 28, 29] among many others. Below we discuss the results most relevant to the current work.

Original principal curves are simple smooth curves with no self-intersections. In [19], Kégl et al. represented principal curves as polygonal lines, and proposed a regularized version of principal curves. They gave a practical algorithm to compute such a polygonal principal curve. This algorithm was later extended in [18] into a *principal graph* algorithm to compute the skeleton graph of hand-written digits and characters. To the best of our knowledge, this was the first algorithm to explicitly allow self-intersections in the output principal curves. However, this principal graph algo-

rithm could only handle 2D images. Very recently in [22], Ozertem and Erdogmus proposed a new definition for the principal curve associated to the probability density function. Intuitively, imagining the probability density function as a terrain, their principal curves are the mountain ridges. A rigorous definition can be made in terms of the Hessian of the probability density. Their approach has several nice properties, including connections to the popular mean-shift clustering algorithm. It also allows for certain bifurcations and self-intersections. However, the output of the algorithm is only a collection of points with neither connectivity information, nor the information about which points are junction points (graph nodes) and which points belong to the same arc in the principal graph. Furthermore, the algorithm depends on reliable density estimation from input data, which is a challenging task for high dimensional data.

Aanijaneya et al. [2] recently proposed perhaps the first general algorithm to approximate a hidden metric graph from an input graph with theoretical guarantees. While the goal of [2] is to approximate a metric graph, their algorithm can also be used to skeletonize data. The algorithm relies on inspecting the local neighborhood of each point to first classify whether it should be a "branching point" or an "edge point". Although this approach has theoretical guarantees when the sampling is nice and the parameters are chosen correctly, it is often hard to find suitable parameters in practice, and such local decisions tend to be less reliable when the input data are not as nice (such as a "fat" junction region). In the section on experimental results we show that our algorithm tends to be more robust in practical applications.

Finally we note that the concept of the Reeb graph has been used in a number of applications in graphics, visualization, and computer vision (see [6] for a survey). However, it has been typically used with mesh structures rather than a tool for analyzing unorganized point cloud data, especially in high dimensions, where constructing meshes is prohibitively expensive. An exception is the very recent work[20], where the authors propose to use the Reeb graph for point cloud data and show applications for several data-sets still in 3D. The advantage of our approach is that it is based on the Rips complex, which allows for a general and cleaner Reeb graph reconstruction algorithm with theoretical justification (see [9, 15] and Theorem 3.1).

## 2  Reeb Graphs

We now give a very brief description of the Reeb graph; see Section VI.4 of [12] for a more formal discussion of it. Let $f : X \to \mathbb{R}$ be a continuous function defined on a domain $X$. For each scalar value $a \in \mathbb{R}$, the level set $f^{-1}(a) = \{x \in X \mid f(x) = a\}$ may have multiple connected components. The *Reeb graph* of $f$, denoted by $\mathcal{R}_f(X)$, is obtained by continuously identifying every connected component in a level set to a single point. In other words, $\mathcal{R}_f(X)$ is the image of a continuous surjective map $\Phi : X \to \mathcal{R}_f(X)$ where $\Phi(x) = \Phi(y)$ *if and only if* $x$ and $y$ come from the same connected component of a level set of $f$.

Intuitively, as the value $a$ increases, connected components in the level set $f^{-1}(a)$ appear, disappear, split and merge, and the Reeb graph of $f$ tracks such changes. The Reeb graph is an abstract graph. Its nodes indicate changes in the connected components in level sets, and each arc represents the evolution of a connected component before it is merged, killed, or split. See the right figure for an example, where we show (an embedding of) the Reeb graph of the height function $f$ defined on a topological torus. The Reeb graph $\mathcal{R}_f(X)$ provides a simple yet meaningful abstraction of the input domain $X$ w.r.t function $f$.

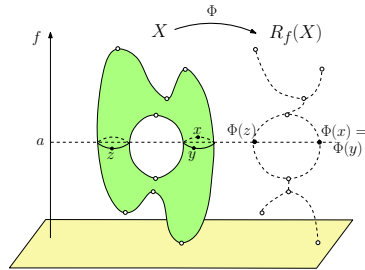

**Computation in discrete setting.**   Assume the input domain is modeled by a simplicial complex $K$. Specifically, a *k-dimensional simplex* $\sigma$ is simply the convex combination of $k + 1$ independent points $\{v_0, \ldots, v_k\}$, and any simplex formed by a subset of its vertices is called a *face* of $\sigma$. A *simplical complex K* is a collection of simplices with the property that if a simplex $\sigma$ is in $K$, then any face of it is also in $K$. A *piecewise-linear (PL) function* $f$ defined on $K$ is a function with values given at vertices of $K$ and linearly interpolated within each simplex in $K$. Given a PL-function $f$ on $K$, its Reeb graph $\mathcal{R}_f(K)$ is decided by all the 0, 1 and 2-simplices from $K$, which are the vertices, edges, and triangles of $K$. Hence from now on we use only 2-dimensional simplicial complex.

Given a PL function defined on a simplicial complex domain $K$, its Reeb graph can be computed efficiently in $O(n \log n)$ expected time by a simple randomized algorithm [15], where $n$ is the size of $K$. In fact, the algorithm outputs the so-called *augmented Reeb graph $R$*, which contains the image of all vertices in $K$ under the surjection map

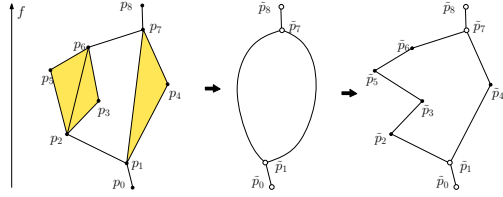

$\Phi : K \to R$ introduced earlier. See figure on the right: the Reeb graph (middle) is an abstract graph with four nodes, while the augmented Reeb graph (on the right) shows the image of all vertices (i.e, $\tilde{p}_i$s). From the augmented Reeb graph $R$, we can easily extract junction points (graph nodes), the set of points from the input data that should be mapped to each graph arc, as well as the connectivity between these points along the Reeb graph (e.g, $\tilde{p}_1 \tilde{p}_4 \tilde{p}_7$ form one arc between $\tilde{p}_1$ and $\tilde{p}_7$).

# 3 Method

## 3.1 Basic algorithm

**Step 1: Set up complex K.** The input data we consider can be a set of points sampled from a hidden domain or a probabilistic distribution, or it can be the distance matrix, or simply the proximity graph, among a set of points. (So the input points do not have to be embedded.) Our goal is to compute (possibly an embedding of) a skeleton graph from the input data. First, we construct an appropriate space approximating the hidden domain that input points are sampled from. We use a simplicial complex K to model such a space.

Specifically, given input sampled points $P$ and the distance matrix of $P$, we first construct a proximity graph based on either $r$-neighborhood or $k$-nearest neighbors(NN) information; that is, a point $p \in P$ is connected either to all its neighbors within $r$ distance to $p$, or to its $k$-NNs. We add all points in $P$ and all edges from this proximity graph to the simplicial complex K we are building. Next, for any three vertices $p_1, p_2, p_3 \in P$, if they are pairwise connected in the proximity graph, we add the triangle $\triangle p_1 p_2 p_3$ to K. Note that if the proximity graph is already given as the input, then we simply fill in a triangle whenever all its three edges are in the proximity graph to obtain the target simplicial complex K. We remark that there is only one parameter involved in the basic algorithm, which is the parameter $r$ (if we use $r$-neighborhood) or $k$ (if we use $k$-NN) to specify the scale with which we look at the input data.

*Motivation behind this construction.* If the proximity graph is built based on $r$-neighborhood, then the above construction is simply that of the so-called *Vietoris-Rips complex*, which has been widely used in manifold reconstruction (especially surface reconstruction) community to recover the hidden domain from its point samples. Intuitively, imagine that we grow a ball of radius $r$ around each sample point. The union of these balls roughly captures the hidden domain at scale $r$. On the other hand, the topological structure of the union of these balls is captured by the so-called *Čech complex*, which mathematically is the nerve of this union of balls. Hence the Čech complex captures the topology of the hidden domain when the sampling is reasonable (see e.g., [8, 21]). However, Čech complex is hard to compute, and the Vietoris-Rips complex is a practical approximation of the Čech complex that is much easier to construct. Furthermore, it has been shown that the Reeb graph of a hidden manifold can be approximated with theoretical guarantees from the Rips complex [9].

**Step 2: Reeb graph computation.** Now we have a simplicial complex K that approximates the hidden domain. In order to extract the skeleton graph using the Reeb graph, we need to define a function g on K that respects its shape. It is also desirable that this function is intrinsic, given that input points may not be embedded. To this end, we construct the function g as the geodesic distance in K to a certain base point $\mathbf{b} \in K$. We compute the base point by taking an arbitrary point $v \in K$ and choosing $\mathbf{b}$ as the point furtherest away from $v$. Intuitively, this base point is an extreme point. If

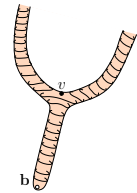

the underlying domain indeed has a branching filamentary structure, then the geodesic distance to $\mathbf{b}$ tends to progress along each filament, and branch out at junction points. See the right figure for an example, where the thin curves are level sets of the geodesic distance function to the base point $\mathbf{b}$.

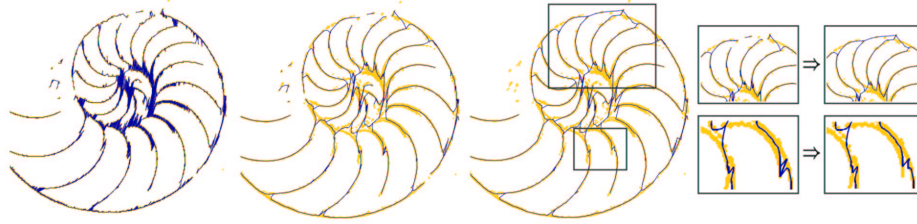

Figure 1: Overview of the algorithm. The input points are light (yellow) shades beneath dark curves. (Left): the augmented Reeb graph output by our algorithm. (Center): after iterative smoothing. (Right): final output after repairing missing links (e.g top box) and simplification (lower box).

Since the Reeb graph tracks the evolution of the connected components in the level sets, a branching (splitting in the level set) will happen when the level set passes through point $v$.

In our algorithm, the geodesic distance function g to b in K is approximated by the shortest distance in the proximity graph (i.e, the set of edges in K) to b. We then perform the algorithm from [15] to compute the Reeb graph of K with respect to g, and denote the resulting Reeb graph as R. Recall that this algorithm in fact outputs the augmented Reeb graph R. Hence we not only obtain a graph structure, but also the set of input points (together with their connectivity) that are mapped to every graph arc in this graph structure.

**Time complexity.** The time complexity of the basic algorithm is the summation of time to compute (A) the proximity graph, (B) the complex K from the proximity graph, (C) the geodesic distance and (D) the Reeb graph. (A) is $O(n^2)$ for high dimensional data (and can be made near-linear for data in very low dimensions) where $n$ is the number of input points. (B) is $O(k^3 n)$ if each point takes $k$ neighbors. (C) and (D) takes time $O(m \log n) = O(k^3 n \log n)$ where $m$ is the size of K. Hence overall, the time complexity is $O(n^2 + k^3 n \log n)$. For high dimensional data sets, this is dominated by the computation of the proximity graph $O(n^2)$.

**Theoretical guarantees.** Given a domain $X$ and a function $f : X \to \mathbb{R}$ defined on it, the topology (i.e, the number of independent loops) of the Reeb graph $\mathcal{R}_f(X)$ may not reflect that of the given domain $X$. However, in our case, we have the following result which offers a partial theoretical guarantee for the basic algorithm. Intuitively, the theorem states that if the hidden space is a graph $G$, and if our simplicial complex K approximates $G$ both in terms of topology (as captured by homotopy equivalence) and metric (as captured by the $\varepsilon$-approximation), then the Reeb graph captures all loops in $G$. Below, $d_Y(\cdot, \cdot)$ denotes the geodesic distance in domain $Y$.

**Theorem 3.1** *Suppose K is homotopy equivalent to a graph $G$, and $h$ : K $\to G$ is the corresponding homotopy. Assume that the metric is $\varepsilon$-approximated under $h$; that is, $|d_K(x, y) - d_G(h(x), h(y))| \leq \varepsilon$ for any $x, y \in$ K, Let $R$ be the Reeb graph of K w.r.t the geodesic distance function to an arbitrary base point $\mathbf{b} \in$ K. If $\varepsilon < l/4$, where $l$ is the length of the shortest arc in $G$, we have that there is a one-to-one correspondence between loops in $R$ and loops in $G$.*

The proof can be found in the full version [13]. It relies on results and observations from [9]. The above result can be made even stronger: (i) There is not only a one-to-one correspondence between loops in $R$ and in $G$, the ranges of each pair of corresponding loops are also close. Here, the *range* of a loop $\gamma$ w.r.t. a function $f$ is the interval $[\min_{x \in \gamma} f(x), \max_{x \in \gamma} f(x)]$. (ii) The condition on $\varepsilon < l/4$ can be relaxed. Furthermore, even when $\varepsilon$ does not satisfy this condition, the reconstructed Reeb graph $R$ can still preserve all loops in $G$ whose range is larger than $2\varepsilon$.

## 3.2 Embedding and Visualization

The Reeb graph is an abstract graph. To visualize the skeleton graph, we need to embed it in a reasonable way that reflects the geometry of hidden domain. To this end, if points are not already embedded in 2D or 3D, we project the input points $P$ to $\mathbb{R}^3$ using any standard dimensionality reduction algorithm. We then connect projected points based on their connectivity given in the

augmented Reeb graph R. Each arc of the Reeb graph is now embedded as a polygonal curve. To further improve the quality of this curve, we fix its endpoints, and iteratively smooth it by repeatedly assigning a point's position to be the average of its neighbors' positions. See Figure 1 for an example.

### 3.3  Further Post-processing

In practice, data can be noisy, and there may be spurious branches or loops in the Reeb graph R constructed no matter how we choose parameter $r$ or $k$ to decide the scale. Following [3], there is a natural way to define "features" in a Reeb graph and measure their "importance". Specifically, given a function $f : X \to \mathbb{R}$, imagine we plot its Reeb graph $\mathcal{R}_f(X)$ such that the height of each point $z \in \mathcal{R}_f(X)$ is the function value of all those points in $X$ mapped to $z$. Now we sweep the Reeb graph bottom-up in increasing order of the function values. As we sweep through a point $z$, we inspect what happens to the part of Reeb graph that we already swept, denoted by $\mathcal{R}_f^z := \{w \in \mathcal{R}_f(X) \mid f(w) \le f(z)\}$. When we sweep past a down-fork saddle $s$, there are two possibilities:

(i). The two branches merged by $s$ belong to different connected components, say $C_1$ and $C_2$, in $\mathcal{R}_f^s$. In such case, we have a *branch-feature*, where two disjoint lower-branches in $\mathcal{R}_f^s$ will be merged at $s$. The importance of this feature is the smaller height of the lower-branches being merged. Intuitively, this is the amount we have to perturb function $f$ in order to remove this branch-feature. See the right figure, where the height $h$ of $C_2$ is the importance of this branch-feature. 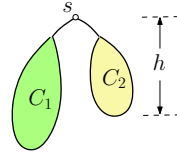

(ii). The two branches merged by $s$ are already connected below $s$ in $\mathcal{R}_f^s$. In such case, when $s$ connects them again, we create a family of new loops. This is called a *loop-feature*. Its size is measured as smallest height of any loop formed by $s$ in $\mathcal{R}_f^s$, where the *height* of a loop $\gamma$ is defined as $\max_{z \in \gamma} f(z) - \min_{z \in \gamma} f(z)$. See the right figure, where the dashed loop $\gamma$ is the thinnest loop created by $s$. 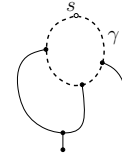

Now if we sweep $\mathcal{R}_f(X)$ top-down, we will also obtain branch-features and loop-features captured by up-fork saddles in a symmetric manner. It turns out that these features (and their sizes) correspond to the so-called *extended persistence* of the Reeb graph $\mathcal{R}_f(X)$ with respect to function $f$ [12]. The size of each feature is called its *persistence*, as it indicates how persistent this feature is as we perturb the function $f$. These features and their persistence can be computed in $O(n \log^2 n)$ time, where $n$ is the number of nodes and arcs in the Reeb graph [3]. We can now simplify the Reeb graph by merging features whose persistence value is smaller than a given threshold. This simplification step not only removes noise, but can also be used as a way to look at features at larger scales.

Finally, there may also be missing data causing missing links in the constructed skeleton graph. Hence in post-processing the user can also choose to first fill some missing links before the simplification step. This is achieved by connecting pairs of degree-1 nodes $(x, y)$ in the Reeb graph whose distances $d(x, y)$ is smaller than certain distance threshold. Here $d(x, y)$ is the input distance between $x$ and $y$ (if the input points are embedded, or the distance matrix is given), not the distance in the simplicial complex K constructed by our algorithm. Connecting $x$ and $y$ may either connect two disjoint component in the Reeb graph, thus creating new branch-features; or form new loop-features. See Figure 1. We do not check the size of the new features created when connecting pairs of vertices. Small newly-created features will be removed in the subsequent simplification step.

## 4  Experimental Results

In this section we first provide comparisons of our algorithm to three existing methods. We then present three sets of experiments to demonstrate the effectiveness of our software and show potential applications of skeleton graph extraction for data analysis.

**Experimental Comparisons.**  We compare our approach with three existing comparable algorithms: (1) the principal graph algorithm (PGA) [18]; (2) the local-density principal curve algorithm (LDPC) [22]; and (3) the metric-graph reconstruction algorithm (MGR) [2]. Note that PGA only works for 2D images. LDPC only outputs *point cloud* at the center of the input data with no connectivity information.

In the figure on the right, we show the skeleton graph of the image of a hand-written Chinese character. Our result is shown in (a). PGA [18] is shown in (b), while the output of (the KDE version of) LDPC [22] is shown in (c). We see that the algorithm from [18], specifically designed for these 2-D applications provides the best output. However, the results of our algorithm, which is completely generic, are

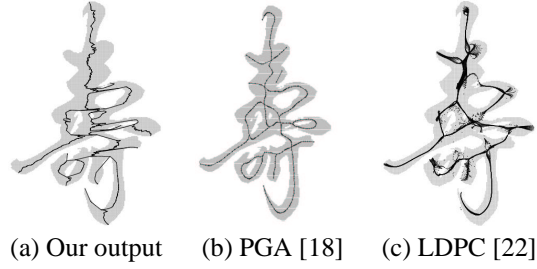

(a) Our output    (b) PGA [18]    (c) LDPC [22]

comparable. On the other hand, the output of LDPC is a point cloud (rather than a graph). In this example, many points do not belong to the 1D structure[1]. We do not show the results from MGR [2] as we were not able to produce a satisfactory result for this data using MGR even after tuning the parameters. However, note that the goal of their algorithm is to approximate a graph metric, which is different from extracting a skeleton graph.

For the second set of comparisons we build a skeleton graph out of an input metric graph. Note that PGA and LDPC cannot handle such graph-type input, and the only comparable algorithm is MGR [2]. We use the image-web

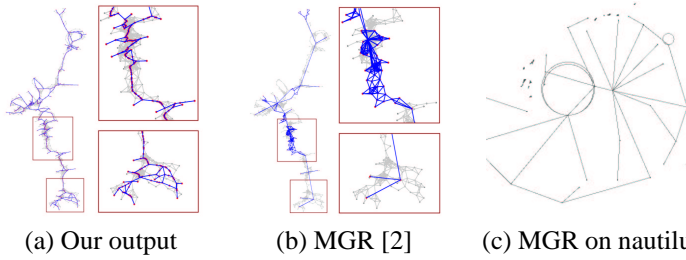

(a) Our output    (b) MGR [2]    (c) MGR on nautilus

data previously used in [2]. Figure (a) on the right is our output and (b) is the output by MGR [2]. The input image web graph is shown in light (gray) color in the background. Finally to provide an additional comparison we apply MGR to image edge detection: (c) above shows the reconstructed edges for the nautilus image used earlier in Figure 1. To be fair, MGR does not provide an embedding, so we should focus on comparing graph structure. Still, MGR collapses the center of nautilus into a single point, while out algorithm is able to recover the structure more accurately[2].

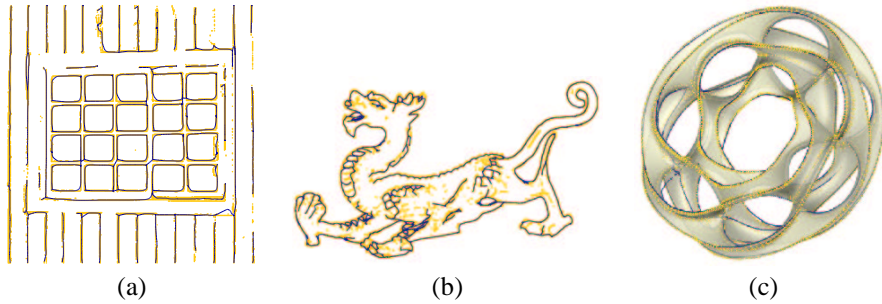

(a)        (b)        (c)

Figure 2: (a) & (b): Edge detection in images. (c) Sharp feature curves detection.

We now proceed with three examples of our algorithms applied to different datasets.

**Example 1: Image edge detection and surface feature curve reconstruction.** In edge detection from images, it is often easy to identify (disconnected) points potentially lying on an edge. We can then use our Reeb-graph algorithm to connect them into curves. See Figure 1 and 2 (a), (b) for some examples. The yellow (light) shades are input potential edge-points computed by a standard edge-detection algorithm based on Roberts edge filter. Original images are given in the full version [13]. In Figure 2 (c), we are given a set of points sampled from a hidden surface model (gray points), and the goal is to extract (sharp) feature curves from it automatically. We first identify points lying around sharp feature lines using a local differential operator (yellow points) and apply our algorithm to connect them into feature lines/graphs (dark curves).

**Example 2: Speech data.**   The input speech data contains utterances of single digits by different speakers. Each utterance is sampled every 10msec with a moving frame. Each sample is represented by the first 13 coefficients resulting from running the standard Perceptual Linear Prediction (PLP) algorithm on the wave file. Given this setup, each utterance of a digit is a trajectory in this 13D feature space.

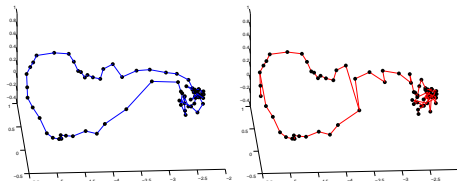

In the left panel, we show the trajectory of an utterance of digit '1' projected to $\mathbb{R}^3$. The right panel shows the graph reconstructed by our algorithm by treating the input simply as a set of points (i.e, removing the time sequence information). No post-processing is performed. Note the main portion of the utterance (the large loop) is well-reconstructed. The cluster of points in the right side corresponds to sampling of silence at the beginning and end of the utterance. This indicates that our algorithm can potentially be used to automatically reconstruct trajectories when the time information is lost.

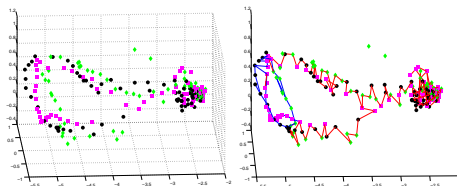

Next, we combine three utterances of the digit '1' and construct the graph from the resulting point cloud shown in the left panel. Each color represents the point cloud coming from one utterance of '1'. As shown in the right panel, the graph reconstructed by our algorithm automatically aligns these three utterances (curves) in the feature space: well-aligned subcurves are merged into single pieces along the graph skeleton, while divergent portions will appear as branches and loops in the graph (see the loops on the left-side of this picture). We expect that our methods can be used to produce a summary representation for multiple similar trajectories (low and high-dimensional curves), to both align trajectories with no time information and to discover convergent and divergent portions of the trajectories.

**Example 3: Molecular simulation.**   The input is a molecular simulation data using the replica-exchange molecular dynamics method [23]. It contains 250K protein conformations, generated by 20 simulation runs, each of which produces a trajectory in the protein conformational space.

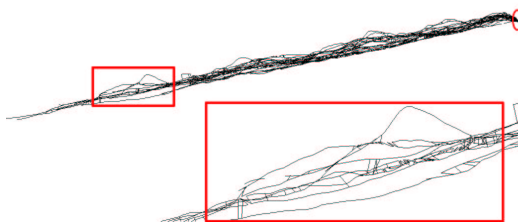

The figure on the right shows a 3D-projection of the Reeb graph constructed by our algorithm. Interestingly, filaments structures can be seen at the beginning of the simulation, which indicates the 20 trajectories at high energy level. As the simulation proceeds, these different simulation runs converge and about $40\%$ of the data points are concentrated in the oval on the right of the figure, which correspond to low-energy conformations. Ideally, simulations at low energy should provide a good sampling in the protein conformational space around the native structure of this protein. However, it turns out that there are several large loops in the Reeb graph close to the native structure (the conformation with lowest energy). Such loop features could be of interest for further investigation.

**Combining with principal curve algorithms.** Finally, our algorithm can be used in combination with principal curve algorithms. In particular, one way is to use our algorithm to first decompose the input data into different arcs of a graph structure, and then use a principal curve algorithm to compute an embedding of this arc in the center of points contributing to it. Alternatively, we can first use the LDPC algorithm [22] to move points to the center of the data, and then perform our algorithm to connect them into a graph structure. Some preliminary results on such combination applied to the hand-written Chinese character can be found in the full version [13].

**Acknowledgments.**   The authors thank D. Chen and U. Ozertem for kindly providing their software and for help with using the software. This work was in part supported by the NSF under CCF-0747082, CCF-1048983, CCF-1116258, IIS-1117707, IIS-0643916.

## Footnotes

[1] Larger $\sigma$ for kernel density estimation fixes that problem but causes important features to disappear.

[2]Tuning the parameters of MGR does not seem to help, see the full version [13] for details.

# References

[1] Open street map. http://www.openstreetmap.org/.

[2] M. Aanjaneya, F. Chazal, D. Chen, M. Glisse, L. Guibas, and D. Morozov. Metric graph reconstruction from noisy data. In *Proc. 27th Sympos. Comput. Geom.*, 2011.

[3] P. K. Agarwal, H. Edelsbrunner, J. Harer, and Y. Wang. Extreme elevation on a 2-manifold. *Discrete and Computational Geometry (DCG)*, 36(4):553–572, 2006.

[4] M. Belkin and P. Niyogi. Laplacian Eigenmaps for dimensionality reduction and data representation. *Neural Comp*, 15(6):1373–1396, 2003.

[5] P. Bendich, B. Wang, and S. Mukherjee. Local homology transfer and stratification learning. In *ACM-SIAM Sympos. Discrete Alg.*, 2012. To appear.

[6] S. Biasotti, D. Giorgi, M. Spagnuolo, and B. Falcidieno. Reeb graphs for shape analysis and applications. *Theor. Comput. Sci.*, 392:5–22, February 2008.

[7] K. Chang and J. Grosh. A unified model for probabilistic principal surfaces. *IEEE Trans. Pattern Anal. Machine Intell.*, 24(1):59–64, 2002.

[8] F. Chazal, D. Cohen-Steiner, and A. Lieutier. A sampling theory for compact sets in Euclidean space. *Discrete Comput. Geom.*, 41(3):461–479, 2009.

[9] T. K. Dey and Y. Wang. Reeb graphs: Approximation and persistence. In *Proc. 27th Sympos. Comput. Geom.*, pages 226–235, 2011.

[10] D Dong and T. J Mcavoy. Nonlinear principal component analysis based on principal curves and neural networks. *Computers & Chemical Engineering*, 20:65–78, 1996.

[11] T. Duchamp and W. Stuetzle. Extremal properties of principal curves in the plane. *The Annals of Statistics*, 24(4):1511–1520, 1996.

[12] H. Edelsbrunner and J. Harer. *Computational Topology, An Introduction*. Amer. Math. Society, 2010.

[13] X. Ge, I. Safa, M. Belkin, and Y. Wang. Data skeletonization via Reeb graphs, 2011. Full version at www.cse.ohio-state.edu/~yusu.

[14] G. Haro, G. Randall, and G. Sapiro. Translated poisson mixture model for stratification learning. *International Journal of Computer Vision*, 80(3):358–374, 2008.

[15] W. Harvey, Y. Wang, and R. Wenger. A randomized $O(mlogm)$ time algorithm for computing Reeb graphs of arbitrary simplicial complexes. In *Proc. 26th Sympos. Comput. Geom.*, pages 267–276, 2010.

[16] T. J. Hastie. *Principal curves and surfaces*. PhD thesis, stanford university, 1984.

[17] T. J. Hastie and W. Stuetlze. Principal curves. *Journal of the American Statistical Association*, 84(406):502–516, 1989.

[18] B. Kégl and A. Krzyżak. Piecewise linear skeletonization using principal curves. *IEEE Trans. Pattern Anal. Machine Intell.*, 24:59–74, January 2002.

[19] B. Kégl, A. Krzyzak, T. Linder, and K. Zeger. Learning and design of principal curves. *IEEE Trans. Pattern Anal. Machine Intell.*, 22:281–297, 2000.

[20] M. Natali, S. Biasotti, G. Patanè, and B. Falcidieno. Graph-based representations of point clouds. *Graphical Models*, 73(5):151 – 164, 2011.

[21] P. Niyogi, S. Smale, and S. Weinberger. Finding the homology of submanifolds with high confidence from random samples. *Discrete Comput. Geom.*, 39(1-3):419–441, 2008.

[22] U. Ozertem and D. Erdogmus. Locally defined principal curves and surfaces. *Journal of Machine Learning Research*, 12:1249–1286, 2011.

[23] I.-H. Park and C. Li. Dynamic ligand-induced-fit simulation via enhanced conformational samplings and ensemble dockings: A survivin example. *J. Phys. Chem. B.*, 114:5144–5153, 2010.

[24] S. T. Roweis and L. K. Saul. Nonlinear dimensionality reduction by locally linear embedding. *Science*, 290(5500):2323–2326, 2000.

[25] B. Scholkopf, A. Smola, and K.R. Muller. Nonlinear Component Analysis as a Kernel Eigenvalue Problem. *Neural Computation*, 10:1299–1319, 2000.

[26] Derek Stanford and Adrian E. Raftery. Finding curvilinear features in spatial point patterns: Principal curve clustering with noise. *IEEE Trans. Pattern Anal. Machine Intell.*, 22(6):601–609, 2000.

[27] J. B. Tenenbaum, V. de Silva, and J. C. Langford. A global geometric framework for nonlinear dimensionality reduction. *Science*, 290(5500):2319–2323, 2000.

[28] R. Tibshirani. Principal curves revisited. *Statistics and Computing*, 2:183–190, 1992.

[29] J. J. Verbeek, N. Vlassis, and B. Kröse. A $k$-segments algorithm for finding principal curves. *Pattern Recognition Letters*, 23(8):1009–1017, 2002.

